# Agglomerative Multivariate Information Bottleneck

**Noam Slonim    Nir Friedman    Naftali Tishby**

School of Computer Science & Engineering, Hebrew University, Jerusalem 91904, Israel

{noamm, nir, tishby }@cs.huji.ac.il

## Abstract

The *information bottleneck* method is an unsupervised model independent data organization technique. Given a joint distribution $P(A, B)$, this method constructs a new variable $T$ that extracts partitions, or clusters, over the values of $A$ that are informative about $B$. In a recent paper, we introduced a general principled framework for multivariate extensions of the information bottleneck method that allows us to consider multiple systems of data partitions that are inter-related. In this paper, we present a new family of simple agglomerative algorithms to construct such systems of inter-related clusters. We analyze the behavior of these algorithms and apply them to several real-life datasets.

## 1  Introduction

The *information bottleneck* (IB) method of Tishby et al [14] is an unsupervised non-parametric data organization technique. Given a joint distribution $P(A, B)$, this method constructs a new variable $T$ that represents partitions of $A$ which are (locally) maximizing the mutual information about $B$. In other words, the variable $T$ induces a *sufficient partition*, or informative features of the variable $A$ with respect to $B$. The construction of $T$ finds a tradeoff between the information about $A$ that we try to minimize, $I(T; A)$, and the information about $B$ which we try to maximize, $I(T; B)$. This approach is particularly useful for co-occurrence data, such as words and documents [12], where we want to capture what information one variable (e.g., use of a word) contains about the other (e.g., the document).

In a recent paper, Friedman et al. [4] introduce *multivariate* extension of the *IB* principle. This extension allows us to consider cases where the data partition is relevant with respect to several variables, or where we construct several systems of clusters simultaneously. In this framework, we specify the desired interactions by a pair of Bayesian networks. One network, $G_{in}$, represents which variables are compressed versions of the observed variables – each new variable compresses its parents in the network. The second network, $G_{out}$, defines the statistical relationship between these new variables and the observed variables that should be maintained.

Similar to the original IB, in Friedman et al. we formulated the general principle as a tradeoff between the (multi) information each network carries. On the one hand, we want to minimize the information maintained by $G_{in}$ and on the other to maximize the information maintained by $G_{out}$. We also provide a characterization of stationary points in this tradeoff as a set of *self-consistent* equations. Moreover, we prove that iterations of these equations converges to a (local) optimum. Then, we describe a *deterministic annealing* procedure

that constructs a solution by tracking the bifurcation of clusters as it traverses the tradeoff curve, similar to the original *IB* method.

In this paper, we consider an alternative approach to solving multivariate *IB* problems which is motivated by the success of the *agglomerative IB* of Slonim and Tishby [11]. As shown there, a bottom-up greedy agglomeration is a simple heuristic procedure that can yield good solutions to the original *IB* problem. Here we extend this idea in the context of multivariate *IB* problems. We start by analyzing the cost of agglomeration steps within this framework. This both elucidates the criteria that guides greedy agglomeration and provides for efficient *local* evaluation rules for agglomeration steps. This construction results with a novel family of information theoretic agglomerative clustering algorithms, that can be specified using the graphs $G_{in}$ and $G_{out}$. We demonstrate the performance of some of these algorithms for document and word clustering and gene expression analysis.

## 2 Multivariate Information Bottleneck

A *Bayesian network structure* $G$ is a DAG that specifies interactions among variables [8]. A distribution $P$ is *consistent* with $G$ (denoted $P \models G$), if $P(X_1, \ldots, X_n) = \prod_i P(X_i \mid \mathbf{Pa}_{X_i}^G)$, where $\mathbf{Pa}_{X_i}^G$ are the parents of $X_i$ in $G$. Our main interest is in the information that the variables $X_1, \ldots, X_n$ contain about each other. A quantity that captures this is the *multi-information* given by

$$\mathcal{I}(X_1, \ldots, X_n) = \mathcal{D}(P(X_1, \ldots, X_n) \| P(X_1) \cdots P(X_n)) \ ,$$

where $\mathcal{D}(p \| q)$ is the familiar Kullback-Liebler divergence [2].

**Proposition 2.1** [4] *Let $G$ be a DAG over* $\{X_1, \ldots, X_n\}$, *and let* $P \models G$ *be a distribution. Then,* $\mathcal{I}^G \equiv \mathcal{I}(X_1, \ldots, X_n) = \sum_i I(X_i; \mathbf{Pa}_{X_i}^G)$.

That is, the multi-information is the sum of *local* mutual information terms between each variable and its parents (denoted $\mathcal{I}^G$).

Friedman et al. define the multivariate *IB* problem as follows. Suppose we are given a set of observed variables, $\mathbf{X} = \{X_1, \ldots, X_n\}$ and their joint distribution $P(X_1, \ldots, X_n)$. We want to "construct" new variables $\mathbf{T}$, where the relations between the observed variables and these new compression variables are specified using a DAG $G_{in}$ over $\mathbf{X} \cup \mathbf{T}$ where the variables in $\mathbf{T}$ are leafs. Thus, each $T_j$ is a stochastic function of a set of variables $\mathbf{U}_j = \mathbf{Pa}_{T_j}^{G_{in}} \subseteq \mathbf{X}$. Once these are set, we have a joint distribution over the combined set of variables: $P(\mathbf{X}, \mathbf{T}) = P(\mathbf{X}) \prod_j P(T_j \mid \mathbf{U}_j)$.

The "relevant" information that we want to preserve is specified by another DAG, $G_{out}$. This graph specifies, for each $T_j$ which variables it predicts. These are simply its children in $G_{out}$. More precisely, we want to predict each $X_i$ (or $T_j$) by $\mathbf{V}_{X_i} = \mathbf{Pa}_{X_i}^{G_{out}}$ (resp. $\mathbf{V}_{T_j} = \mathbf{Pa}_{T_j}^{G_{out}}$), its parents in $G_{out}$. Thus, we think of $\mathcal{I}^{G_{out}}$ as a measure of how much information the variables in $\mathbf{T}$ maintain about their target variables.

The Lagrangian can then be defined as

$$\mathcal{L}[p(T_1 \mid \mathbf{U}_1), \ldots, p(T_k \mid \mathbf{U}_k)] = \mathcal{I}^{G_{out}} - \beta^{-1} \cdot \mathcal{I}^{G_{in}} \ , \tag{1}$$

with a *tradeoff parameter (Lagrange multiplier)* $\beta$. [1] The variation is done subject to the normalization constraints on the partition distributions. Thus, we balance between the information $\mathbf{T}$ loses about $\mathbf{X}$ in $G_{in}$ and the information it preserves in $G_{out}$.

Friedman et al. [4] show that stationary points of this Lagrangian satisfy a set of self-consistent equations. Moreover, they show that iterating these equations converges to a

stationary point of the tradeoff. Then, extending the procedure of Tishby et al [14], they propose a procedure that searches for a solution of the *IB* equations using a 'deterministic annealing' approach [9]. This is a top-down hierarchical algorithm that starts from a single cluster for each $T_j$ at $\beta \to 0$, and then undergoes a cascade of cluster splits as $\beta$ is being "cooled". These determines "soft" trees of clusters (one for each $T_j$) that describe solutions at different tradeoff values of $\beta$.

## 3  The Agglomerative Procedure

For the original *IB* problem, Slonim and Tishby [11] introduced a simpler procedure that performs greedy bottom-up merging of values. Several successful applications of this algorithm are already presented for a variety of real-world problems [10, 12, 13, 15]. The main focus of the current work is in extending this approach for the multivariate *IB* problem. As we will show, this will lead to further insights about the method, and also provide a rather simple and intuitive clustering procedures.

We consider procedures that start with a set of clusters for each $T_j$ (usually the most fine-grained solution we can consider where $T_j = \mathbf{U}_j$) and then iteratively reduce the cardinality of one of the $T_j$'s by *merging* two values $t_j^l$ and $t_j^r$ of $T_j$ into a single value $\bar{t}_j$. To formalize this notion we must define the membership probability of a new cluster $\bar{t}_j$, resulting from merging $\{t_j^l, t_j^r\} \Rightarrow \bar{t}_j$ in $T_j$. This is done rather naturally by

$$p(\bar{t}_j \mid \mathbf{U}_j) = p(t_j^l \mid \mathbf{U}_j) + p(t_j^r \mid \mathbf{U}_j) \ . \tag{2}$$

In other words, we view the event $\bar{t}_j$ as the union of the events $t_j^l$ and $t_j^r$.

Given the membership probabilities, at each step we can also draw the connection between $T_j$ and the other variables. This is done using the following proposition which is based on the conditional independence assumptions given in $G_{in}$.

**Proposition 3.1** *Let* $\mathbf{Y}, \mathbf{Z} \subset \mathbf{X} \cup \mathbf{T} \setminus \{T_j\}$ *then,*

$$p(\mathbf{Y} \mid \bar{t}_j, \mathbf{Z}) = \pi_{l,\mathbf{z}} \cdot p(\mathbf{Y} \mid t_j^l, \mathbf{Z}) + \pi_{r,\mathbf{z}} \cdot p(\mathbf{Y} \mid t_j^r, \mathbf{Z}) \ , \tag{3}$$

*where* $\Pi_{\mathbf{Z}} = \{\pi_{l,\mathbf{z}}, \ \pi_{r,\mathbf{z}}\} = \{\frac{p(t_j^l \mid \mathbf{Z})}{p(\bar{t}_j \mid \mathbf{Z})}, \frac{p(t_j^r \mid \mathbf{Z})}{p(\bar{t}_j \mid \mathbf{Z})}\}$ *is the* merger distribution *conditioned on* $\mathbf{Z}$.

In particular, this proposition allows us to evaluate all the predictions defined in $G_{out}$ and all the informations terms in $\mathcal{L}$ that involve $T_j$.

The crucial question in an agglomerative process is of course which pair to merge at each step. We know that the merger "cost" in our terms is exactly the difference in the values of $\mathcal{L}$, before and after the merger. Let $T_j^{bef}$ and $T_j^{aft}$ denote the random variables that correspond to $T_j$, before and after the merger, respectively. Thus, the values of $\mathcal{L}$ before and after the merger are calculated based on $T_j^{bef}$ and $T_j^{aft}$. The merger cost is then simply given by,

$$\Delta\mathcal{L}(t_j^l, t_j^r) = \mathcal{L}^{bef} - \mathcal{L}^{aft} \ . \tag{4}$$

The greedy procedure evaluates all the potential mergers (for all $T_j$) and then applies the best one (i.e., the one that minimizes $\Delta\mathcal{L}(t_j^l, t_j^r)$). This is repeated until all the variables in $\mathbf{T}$ degenerate into trivial clusters. The resulting set of trees describes a range of solutions at different resolutions.

This agglomerative approach is different in several important aspects from the deterministic annealing approach described above. In that approach, by "cooling" (i.e., increasing) $\beta$, we move along a tradeoff curve from the trivial - single cluster - solution toward solutions with higher resolutions that preserve more information in $G_{out}$. In contrast, in the

agglomerative approach we progress in the opposite direction. We start with a high resolution clustering and as the merging process continues we move toward more and more compact solutions. During this process $\beta$ is kept constant and the driving force is the reduction in the cardinality of the $T_j$'s. Therefore, we are able to look for good solutions in different resolutions for a *fixed* tradeoff parameter $\beta$. Since the merging does not attempt directly to maintain the (stationary) self-consistent "soft" membership probabilities, we do not expect the self-consistent equations to hold at solutions found by the agglomerative procedure. On the other hand, the agglomerative process is much simpler to implement and fully deterministic. As we will show, it provides sufficiently good solutions for the *IB* problem in many situations.

## 4  Local Merging Criteria

In the procedure we outline above, at every step there are $O(|T_j|^2)$ possible mergers of values of $T_j$ (for every $j$). A direct calculation of the costs of all these potential mergers is typically infeasible. However, it turns out that one may calculate $\Delta\mathcal{L}(t_j^l, t_j^r)$ while examining only the probability distributions that involve $t_j^l$ and $t_j^r$ directly. Generalizing the results of [11] for the original *IB*, we now develop a closed-form formula for $\Delta\mathcal{L}(t_j^l, t_j^r)$.

To describe this result we need the following definition. The *Jensen-Shannon (JS) divergence* [7, 3] between two probabilities $p_1, p_2$ is given by

$$JS_\Pi(p_1, p_2) = \pi_1 KL[p_1\|\bar{p}] + \pi_2 KL[p_2\|\bar{p}]$$

where $\Pi = \{\pi_1, \pi_2\}$ is a normalized probability and $\bar{p} = \pi_1 p_1 + \pi_2 p_2$. The $JS$ divergence is equal zero if and only if both its arguments are identical. It is upper bounded and symmetric, though it is not a metric. One interpretation of the $JS$-divergence relates it to the (logarithmic) measure of the likelihood that the two sample distributions originate by the most likely common source, denoted by $\bar{p}$. In addition, we need the notation $\mathbf{V}_{X_i}^{-j} = \mathbf{V}_{X_i} - \{T_j\}$ (similarly for $\mathbf{V}_{T_\ell}^{-j}$).

**Theorem 4.1** *Let $t_j^l, t_j^r \in T_j$ be two clusters. Then, $\Delta\mathcal{L}(t_j^l, t_j^r) = p(\bar{t}_j) \cdot d(t_j^l, t_j^r)$ where*

$$
\begin{aligned}
d(t_j^l, t_j^r) \equiv & \sum_{i: T_j \in \mathbf{V}_{X_i}} E_{P(\cdot|\bar{t}_j)}[JS_{\Pi_{\mathbf{V}_{X_i}^{-j}}}(p(X_i \mid t_j^l, \mathbf{V}_{X_i}^{-j}), p(X_i \mid t_j^r, \mathbf{V}_{X_i}^{-j}))] \\
& + \sum_{\ell: T_j \in \mathbf{V}_{T_\ell}} E_{P(\cdot|\bar{t}_j)}[JS_{\Pi_{\mathbf{V}_{T_\ell}^{-j}}}(p(T_\ell \mid t_j^l, \mathbf{V}_{T_\ell}^{-j}), p(T_\ell \mid t_j^r, \mathbf{V}_{T_\ell}^{-j}))] \\
& + JS_\Pi(p(\mathbf{V}_{T_j} \mid t_j^l), p(\mathbf{V}_{T_j} \mid t_j^r)) - \beta^{-1} \cdot JS_\Pi(p(\mathbf{U}_j \mid t_j^l), p(\mathbf{U}_j \mid t_j^r)) \ .
\end{aligned}
$$

A detailed proof of this theorem will be given elsewhere. Thus, the merger cost is a multiplication of the weight of the merger components ($p(\bar{t}_j)$) with their "distance" given by $d(t_j^l, t_j^r)$. Notice that due to the properties of the $JS$-divergence, this distance is symmetric. In addition, the last term in this distance has the opposite sign to the first three terms. Thus, the distance between two clusters is a tradeoff between these two factors. Roughly speaking, we may say that the distance is minimized for pairs that give similar predictions about the variables connected with $T_j$ in $G_{out}$ and have different predictions (minimum overlap) about the variables connected with $T_j$ in $G_{in}$. We notice also the analogy between this result and the main theorem in [4]. In [4] the optimization is governed by the $KL$ divergences between data and cluster's centroids, or by the likelihood that the data was generated by the centroid distribution. Here the optimization is controlled through the $JS$ divergences, i.e. the likelihood that the two clusters have a common source.

Next, we notice that after applying a merger, only a small portion of the other mergers costs change. The following proposition characterizes these costs.

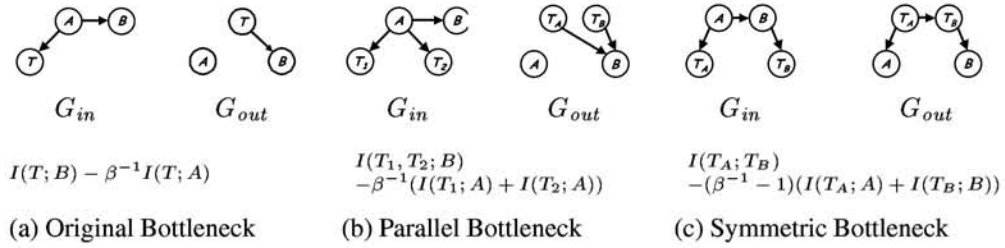

| $G_{in}$ | $G_{out}$ | $G_{in}$ | $G_{out}$ | $G_{in}$ | $G_{out}$ |

$$I(T;B) - \beta^{-1}I(T;A) \qquad \begin{array}{c} I(T_1, T_2; B) \\ -\beta^{-1}(I(T_1; A) + I(T_2; A)) \end{array} \qquad \begin{array}{c} I(T_A; T_B) \\ -(\beta^{-1} - 1)(I(T_A; A) + I(T_B; B)) \end{array}$$

(a) Original Bottleneck     (b) Parallel Bottleneck     (c) Symmetric Bottleneck

Figure 1: The source and target networks and the corresponding Lagrangian for the three examples we consider.

**Proposition 4.2** *The merger $\{t_j^l, t_j^r\} \Rightarrow \bar{t}_j$ in $T_j$ can change the cost $\Delta \mathcal{L}(t_\ell^l, t_\ell^r)$ only if $p(\bar{t}_j, \bar{t}_\ell) > 0$ and $T_j, T_\ell$ co-appear in some information term in $\mathcal{I}^{G_{out}}$.*

This proposition is particularly useful, when we consider "hard" clustering where $T_j$ is a (deterministic) function of $\mathbf{U}_j$. In this case, $p(\bar{t}_j, \bar{t}_\ell)$ is often zero (especially when $T_j$ and $T_\ell$ compressing similar variables, i.e., $\mathbf{U}_j \cap \mathbf{U}_\ell \neq \emptyset$). In particular, after the merger $\{t_j^l, t_j^r\} \Rightarrow \bar{t}_j$, we do not have to reevaluate merger costs of other values of $T_j$, except for mergers of $\bar{t}_j$ with each of these values.

In the case of hard clustering we also find that $I(T_j; \mathbf{U}_j) = H(T_j)$ (where $H(p)$ is Shannon's entropy). Roughly speaking, we may say that $H(p)$ is decreasing for less balanced probability distributions $p$. Therefore, increasing $\beta^{-1}$ will result with a tendency to look for less balanced "hard" partitions and vice verse. This is reflected by the fact that the last term in $d(t_j^l, t_j^r)$ is then simplified through $JS_\Pi(p(\mathbf{U}_j \mid t_j^l), p(\mathbf{U}_j \mid t_j^r)) = H(\Pi)$.

## 5 Examples

We now briefly consider three examples of the general methodology. For brevity we focus on the simpler case of hard clustering. We first consider the example shown in figure 1(a). This choice of graphs results in the *original IB* problem. The merger cost in this case is given by,

$$\Delta \mathcal{L}(t^l, t^r) = p(\bar{t}) \cdot (JS_\Pi(p(B \mid t^l), p(B \mid t^r)) - \beta^{-1}H(\Pi)) \ . \qquad (5)$$

Note that for $\beta^{-1} \to 0$ we get exactly the algorithm presented in [11].

One simple extension of the original *IB* is the *parallel* bottleneck [4]. In this case we introduce two variables $T_1$ and $T_2$ as in Figure 1(b), both of them are functions of $A$. Similarly to the original *IB*, $G_{out}$ specifies that $T_1$ and $T_2$ should predict $B$. We can think of this requirement as an attempt to decompose the information $A$ contains about $B$ into two "orthogonal" components. In this case, the merger cost for $T_1$ is given by,

$$\Delta \mathcal{L}(t_1^l, t_1^r) = p(\bar{t}_1) \cdot (E_{P(\cdot|\bar{t}_1)}[JS_{\Pi_{T_2}}(p(B \mid t_1^l, T_2), p(B \mid t_1^r, T_2))] - \beta^{-1}H(\Pi)) \ . \quad (6)$$

Finally, we consider the *symmetric* bottleneck [4, 12]. In this case, we want to compress $A$ into $T_A$ and $B$ into $T_B$ so that $T_A$ extracts the information $A$ contains about $B$, and at the same time $T_B$ extracts the information $B$ contains about $A$. The DAG $G_{in}$ of figure 1(c) captures the form of the compression. The choice of $G_{out}$ is less obvious and several alternatives are described in [4]. Here, we concentrate only in one option, shown in figure 1(c). In this case we attempt to make each of $T_A$ and $T_B$ sufficient to separate $A$ from $B$. Thus, on one hand we attempt to compress, and on the other hand we attempt to make $T_A$ and $T_B$ as informative about each other as possible. The merger cost in $T_A$ is given by

$$\Delta \mathcal{L}(t_A^l, t_A^r) = p(\bar{t}_A) \cdot JS_\Pi(p(T_B \mid t_A^l), p(T_B \mid t_A^r)) - (\beta^{-1} - 1)H(\Pi) \ , \qquad (7)$$

while for merging in $T_B$ we will get an analogous expression.

## 6 Applications

We examine a few applications of the examples presented above. As one data set we used a subset of the 20 newsgroups corpus [6] where we randomly choose 2000 documents evenly distributed among the 4 science discussion groups (*sci.crypt, sci.electronics, sci.med* and *sci.space*).[2] Our pre-processing included ignoring file headers (and the subject lines), lowering upper case and ignoring words that contained non 'a..z' characters. Given this document set we can evaluate the joint probability $p(W, D)$, which is the probability that a random word position is equal to $w \in W$ and at the same time the document is $d \in D$. We sort all words by their contribution to $I(W; D)$ and used only the 2000 'most informative' ones, ending up with a joint probability with $|W| = |D| = 2000$.

We first used the original *IB* to cluster $W$, while trying to preserve the information about $D$. This was already done in [12] with $\beta^{-1} = 0$, but in this new experiment we took $\beta^{-1} = 0.15$. Recall that increasing $\beta^{-1}$ results in a tendency for finding less balanced clusters. Indeed, while for $\beta^{-1} = 0$ we got relatively balanced word clusters (high $H(T_W)$), for $\beta^{-1} = 0.15$ the probability $p(T_W)$ is much less smooth. For 50 word clusters, one cluster contained almost half of the words, while the other clusters were typically much smaller. Since the algorithm also tries to maximize $I(T_W; D)$, the words merged into the big cluster are usually the less informative words about $D$. Thus, a word must be highly informative to stay out of this cluster. In this sense, increasing $\beta^{-1}$ is equivalent for inducing a "noise filter", that leave only the most informative features in specific clusters. In figure 2 we present $p(D \mid t_W)$ for several clusters $t_W \in T_W$. Clearly, words that passed the "filter" form much more informative clusters about the real structure of $D$. A more formal demonstration of this effect is given in the right panel of figure 2. For a given compression level (i.e. a given $I(T_W; W)$), we see that taking $\beta^{-1} = 0.15$ preserve much more information about $D$.

While an exact implementation of the symmetric *IB* will require alternating mergers in $T_W$ and $T_D$, an approximated approach require only two steps. First we find $T_W$. Second, we project each $d \in D$ into the low dimensional space defined by $T_W$, and use this more robust representation to extract document clusters $T_D$. Approximately, we are trying to find $T_W$ and $T_D$ that will maximize $I(T_W; T_D)$. This two-phase *IB* algorithm was shown in [12] to be significantly superior to six other document clustering methods, when the performance are measured by the correlation of the obtained document clusters with the real newsgroup categories. Here we use the same procedure, but for finding $T_W$ we take $\beta^{-1} = 0.15$ (instead of zero). Using the above intuition we predict this will induce a cleaner representation for the document set. Indeed, the averaged correlation of $T_D$ (for $|T_D| = 4$) with the original categories was 0.65, while for $\beta^{-1} = 0$ it was 0.58 (the average is taken over different number of word clusters, $|T_W| = 10, 11...50$). Similar results were obtained for all the 9 other subsets of the 20 newsgroups corpus described in [12].

As a second data set we used the gene expression measurements of $\sim$ 6800 genes in 72 samples of Leukemia [5]. The sample annotations included type of leukemia (*ALL* vs. *AML*), type of cells, source of sample, gender and donating hospital. We removed genes that were not expressed in the data and normalized the measurements of each sample to get a joint probability $P(G, A)$ over genes and samples (with uniform prior on samples). We sorted all genes by their contribution to $I(G; A)$ and chose the 500 most informative ones, which capture 47% of the original information, ending up with a joint probability with $|A| = 72$ and $|G| = 500$.

We first used an exact implementation of the symmetric *IB* with alternating mergers be-

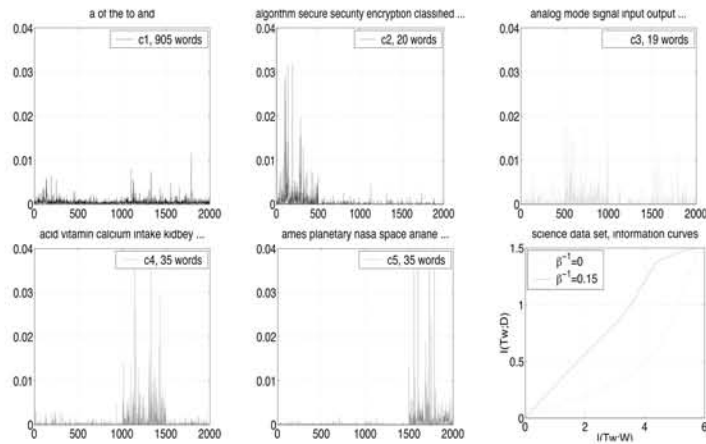

Figure 2: $P(D \mid t_W)$ for 5 word clusters, $t_W \in T_W$. Documents $1 - 500$ belong to *sci.crypt* category, $501 - 1000$ to *sci.electronics*, $1001 - 1500$ to *sci.med* and $1501 - 2000$ to *sci.space*. In the title of each panel we see the 5 most frequent words in the cluster. The 'big' cluster (upper left panel) is clearly less informative about the structure of $D$. In the lower right panel we see the two information curves. Given some compression level, for $\beta^{-1} = 0.15$ we preserve much more information about $D$ than for $\beta^{-1} = 0$.

tween both clustering hierarchies (and $\beta^{-1} = 1$). For $|T_A| = 2$ we found an almost perfect correlation with the *ALL* vs. *AML* annotations (with only 4 exceptions). For $|T_A| = 8$ and $|T_G| = 10$ we found again high correlation between our sample clusters and the different sample annotations. For example, one cluster contained 10 samples that were all annotated as *ALL* type, taken from male patients in the same hospital. Almost all of these 10 were also annotated as T-cells, taken from bone marrow. Looking at $p(T_A \mid T_G)$ we see that given the third genes cluster (which contained 17 genes) the probability of the above specific samples cluster is especially high. Further such analysis might yield additional insights about the structure of this data and will be presented elsewhere.

Finally, to demonstrate the performance of the *parallel IB* we apply it to the same data. Using the *parallel IB* algorithm (with $\beta^{-1} = 0$) we clustered the arrays $A$ into two clustering hierarchies, $T_1$ and $T_2$, that try *together* to capture the information about $G$. For $|T_j| = 4$ we find that each $I(T_j; G)$ preserve about 15% of the original information. However, taking $|T_j| = 2$ (i.e. again, just 4 clusters) we see that the combination of the hierarchies, $I(T_1, T_2; G)$, preserve 21% of the original information. We then compared the two partitions we found against sample annotations. We found that the first hierarchy with $|T_1| = 2$ almost perfectly match the split between B-cells and T-cells (among the 47 samples for which we had this annotation). The second hierarchy, with $|T_2| = 2$ separates a cluster of 18 samples, almost all of which are *ALL* samples taken from the bone marrow of patients from the same hospital. These results demonstrate the ability of the algorithm to extract in parallel different meaningful independent partitions of the data.

## 7  Discussion

The analysis presented by this work enables to implement a family of novel agglomerative clustering algorithms. All of these algorithms are motivated by one variational framework given by the *multivariate IB* method. Unlike most other clustering techniques, this is a principled model independent approach, which aims directly at the extraction of informative structures about given observed variables. It is thus very different from maximum-

likelihood estimation of some mixture model and relies on fundamental information theoretic notions, similar to rate distortion theory and channel coding. In fact the multivariate IB can be considered as a multivariate coding result. The fundamental tradeoff between the compressed multi-information $\mathcal{I}^{G_{in}}$ and the preserved multi-information $\mathcal{I}^{G_{out}}$ provides a generalized coding limiting function, similar to the information curve in the original IB and to the rate distortion function in lossy compression. Despite the only local-optimality of the resulting solutions this information theoretic quantity – the fraction of the multi-information that is extracted by the clusters – provides an objective figure of merit for the obtained clustering schemes.

The suggested approach of this paper has several practical advantages over the 'deterministic annealing' algorithms suggested in [4], as it is simpler, fully deterministic and non-parametric. There is no need to identify cluster splits which is usually rather tricky. Though agglomeration procedures do not scale linearly with the sample size as top down methods do, there exist several heuristics to improve the complexity of these algorithms (e.g. [1]).

While a typical initialization of an agglomerative procedure induces "hard" clustering solutions, all of the above analysis holds for "soft" clustering as well. Moreover, as already noted in [11], the obtained "hard" partitions can be used as a platform to find also "soft" solutions through a process of "reverse annealing". This raises the possibility for using an agglomerative procedure over "soft" clustering solutions, which we leave for future work.

We could describe here only a few relatively simple examples. These examples show promising results on non trivial real life data. Moreover, other choices of $G_{in}$ and $G_{out}$ can yield additional novel algorithms with applications over a variety of data types.

### Acknowledgements

This work was supported in part by the Israel Science Foundation (ISF), the Israeli Ministry of Science, and by the US-Israel Bi-national Science Foundation (BSF). N. Slonim was also supported by an Eshkol fellowship. N. Friedman was also supported by an Alon fellowship and the Harry & Abe Sherman Senior Lectureship in Computer Science.

## Footnotes

[1] Notice that under this formulation we would like to maximize $\mathcal{L}$. An equivalent definition [4] would be to minimize $\mathcal{L} = \mathcal{I}^{G_{in}} - \beta \cdot \mathcal{I}^{G_{out}}$.

[2]We used the same subset already used in [12].

# References

[1] L. D. Baker and A. K. McCallum. Distributional clustering of words for text classification. In *ACM SIGIR 98*.

[2] T. M. Cover and J. A. Thomas. *Elements of Information Theory*. 1991.

[3] R. El-Yaniv, S. Fine, and N. Tishby. Agnostic classification of Markovian sequences. In NIPS'97.

[4] N. Friedman, O. Mosenzon, N. Slonim and N. Tishby Multivariate Information Bottleneck UAI, 2001.

[5] T. Golub, D. Slonim, P. Tamayo, C.M. Huard, J.M. Caasenbeek, H. Coller, M. Loh, J. Downing, M. Caligiuri, C. Bloomfield, and E. Lander. Molecular classification of cancer: class discovery and class prediction by gene expression monitoring *Science* **286**, 531–537, 1999.

[6] K. Lang. Learning to filter netnews. In ICML'95.

[7] J. Lin. Divergence Measures Based on the Shannon Entropy. *IEEE Trans. Info. Theory*, 37(1):145–151, 1991.

[8] J. Pearl. *Probabilistic Reasoning in Intelligent Systems*. 1988.

[9] K. Rose. Deterministic annealing for clustering, compression, classification, regression, and related optimization problems. *Proc. IEEE*, 86:2210–2239, 1998.

[10] N. Slonim, R. Somerville, N. Tishby, and O. Lahav. Objective spectral classification of galaxies using the information bottleneck method. in "Monthly Notices of the Royal Astronomical Society", MNRAS, 323, 270, 2001.

[11] N. Slonim and N. Tishby. Agglomerative Information Bottleneck. In NIPS'99.

[12] N. Slonim and N. Tishby. Document clustering using word clusters via the information bottleneck method. In *ACM SIGIR 2000*.

[13] N. Slonim and N. Tishby. The power of word clusters for text classification. In *ECIR*, 2001.

[14] N. Tishby, F. Pereira, and W. Bialek. The Information Bottleneck method. In *Proc. 37th Allerton Conference on Communication and Computation*. 1999.

[15] N. Tishby and N. Slonim. Data clustering by markovian relaxation and the information bottleneck method. In NIPS'00.
